# Stagewise processing in error-correcting codes and image restoration

**K. Y. Michael Wong**
Department of Physics, Hong Kong University of Science and Technology,
Clear Water Bay, Kowloon, Hong Kong
*phkywong@ust.hk*

**Hidetoshi Nishimori**
Department of Physics, Tokyo Institute of Technology,
Oh-Okayama, Meguro-ku, Tokyo 152-8551, Japan
*nishi@stat.phys.titech.ac.jp*

## Abstract

We introduce stagewise processing in error-correcting codes and image restoration, by extracting information from the former stage and using it selectively to improve the performance of the latter one. Both mean-field analysis using the cavity method and simulations show that it has the advantage of being robust against uncertainties in hyperparameter estimation.

## 1   Introduction

In error-correcting codes [1] and image restoration [2], the choice of the so-called hyperparameters is an important factor in determining their performances. Hyperparameters refer to the coefficients weighing the biases and variances of the tasks. In error correction, they determine the statistical significance given to the parity-checking terms and the received bits. Similarly in image restoration, they determine the statistical weights given to the prior knowledge and the received data. It was shown, by the use of inequalities, that the choice of the hyperparameters is optimal when there is a match between the source and model priors [3]. Furthermore, from the analytic solution of the infinite-range model and the Monte Carlo simulation of finite-dimensional models, it was shown that an inappropriate choice of the hyperparameters can lead to a rapid degradation of the tasks.

Hyperparameter estimation is the subject of many studies such as the "evidence framework" [4]. However, if the prior models the source poorly, no hyperparameters can be reliable [5]. Even if they can be estimated accurately through steady-state statistical measurements, they may fluctuate when interfered by bursty noise sources in communication channels. Hence it is equally important to devise decoding or restoration procedures which are robust against the uncertainties in hyperparameter estimation.

Here we introduce *selective freezing* to increase the tolerance to uncertainties in hy-

perparameter estimation. The technique has been studied for pattern reconstruction in neural networks, where it led to an improvement in the retrieval precision, a widening of the basin of attraction, and a boost in the storage capacity [6]. The idea is best illustrated for bits or pixels with binary states $\pm 1$, though it can be easily generalized to other cases. In a finite temperature thermodynamic process, the binary variables keep moving under thermal agitation. Some of them have smaller thermal fluctuations than the others, implying that they are more certain to stay in one state than the other. This stability implies that they have a higher probability to stay in the correct state for error-correction or image restoration tasks, even when the hyperparameters are not optimally tuned. It may thus be interesting to separate the thermodynamic process into two stages. In the first stage we select those relatively stable bits or pixels whose time-averaged states have a magnitude exceeding a certain threshold. In the second stage we subsequently fix (or freeze) them in the most probable thermodynamic states. Thus these selectively frozen bits or pixels are able to provide a more robust assistance to the less stable bits or pixels in their search for the most probable states.

The two-stage thermodynamic process can be studied analytically in the mean-field model using the cavity method. For the more realistic cases of finite dimensions in image restoration, simulation results illustrate the relevance of the infinite-range model in providing qualitative guidance. Detailed theory of selective freezing is presented in [7].

## 2   Formulation

Consider an information source which generates data represented by a set of Ising spins $\{\xi_i\}$, where $\xi_i = \pm 1$ and $i = 1, \cdots, N$. The data is generated according to the source prior $P_s(\{\xi_i\})$. For error-correcting codes transmitting unbiased messages, all sequences are equally probable and $P_s(\{\xi\}) = 2^{-N}$. For images with smooth structures, the prior consists of ferromagnetic Boltzmann factors, which increase the tendencies of the neighboring spins to stay at the same spin states, that is,

$$P_s(\{\xi\}) \propto \exp\left(\frac{\beta_s}{z}\sum_{\langle ij \rangle} \xi_i \xi_j\right). \tag{1}$$

Here $\langle ij \rangle$ represents pairs of neighboring spins, $z$ is the valency of each site. The data is coded by constructing the codewords, which are the products of $p$ spins $J^0_{i_1 \cdots i_p} = \xi_{i_1} \cdots \xi_{i_p}$ for appropriately chosen sets of of indices $\{i_1, \cdots, i_p\}$. Each spin may appear in a number of $p$-spin codewords; the number of times of appearance is called the valency $z_p$. For conventional image restoration, codewords with only $p = 1$ are transmitted, corresponding to the pixels in the image.

When the signal is transmitted through a noisy channel, the output consists of the sets $\{J_{i_1 \cdots i_p}\}$ and $\{\tau_i\}$, which are the corrupted versions of $\{J^0_{i_1 \cdots i_p}\}$ and $\{\xi_i\}$ respectively, and described by the output probability

$$P_{\text{out}}(\{J\}, \{\tau\}|\{\xi\}) \propto \exp\left(\beta_J \sum J_{i_1 \cdots i_p} \xi_{i_1} \cdots \xi_{i_p} + \beta_\tau \sum \tau_i \xi_i\right). \tag{2}$$

According to Bayesian statistics, the posterior probability that the source sequence is $\{\sigma\}$, given the outputs $\{J\}$ and $\{\tau\}$, takes the form

$$P(\{\sigma\}|\{J\}, \{\tau\}) \propto \exp\left(\beta_J \sum J_{i_1 \cdots i_p} \sigma_{i_1} \cdots \sigma_{i_p} + \beta_\tau \sum \tau_i \sigma_i + \frac{\beta_s}{z}\sum_{\langle ij \rangle} \sigma_i \sigma_j\right). \tag{3}$$

If the receiver at the end of the noisy channel does not have precise information on $\beta_J$, $\beta_\tau$ or $\beta_s$, and estimates them as $\beta$, $h$ and $\beta_m$ respectively, then the $i$th bit of the decoded/restored information is given by $\mathrm{sgn}\langle\sigma_i\rangle$, where

$$\langle\sigma_i\rangle = \frac{\mathrm{Tr}\sigma_i e^{-H\{\sigma\}}}{\mathrm{Tr}e^{-H\{\sigma\}}}, \qquad (4)$$

and the Hamiltonian is given by

$$H\{\sigma\} = -\beta\sum J_{i_1\cdots i_p}\sigma_{i_1}\cdots\sigma_{i_p} - h\sum\tau_i\sigma_i - \frac{\beta_m}{z}\sum_{\langle ij\rangle}\sigma_i\sigma_j. \qquad (5)$$

For the two-stage process of selective freezing, the spins evolve thermodynamically as prescribed in Eq. (4) during the first stage, and the thermal averages $\langle\sigma_i\rangle$ of the spins are monitored. Then we select those spins with $|\langle\sigma_i\rangle|$ exceeding a given threshold $\theta$, and freeze them in the second stage of the thermodynamics. The average of the spin $\tilde{\sigma}_i$ in the second stage is then given by

$$\langle\tilde{\sigma}_i\rangle = \frac{\mathrm{Tr}\tilde{\sigma}_i\prod_j\left[\Theta\left(\langle\sigma_j\rangle^2 - \theta^2\right)\delta_{\tilde{\sigma}_j,\mathrm{sgn}\langle\sigma_j\rangle} + \Theta\left(\theta^2 - \langle\sigma_j\rangle^2\right)\right]e^{-\tilde{H}\{\tilde{\sigma}\}}}{\mathrm{Tr}\prod_j\left[\Theta\left(\langle\sigma_j\rangle^2 - \theta^2\right)\delta_{\tilde{\sigma}_j,\mathrm{sgn}\langle\sigma_j\rangle} + \Theta\left(\theta^2 - \langle\sigma_j\rangle^2\right)\right]e^{-\tilde{H}\{\tilde{\sigma}\}}}, \qquad (6)$$

where $\Theta$ is the step function, $\tilde{H}\{\tilde{\sigma}\}$ is the Hamiltonian for the second stage, and has the same form as Eq. (5) in the first stage. One then regards $\mathrm{sgn}\langle\tilde{\sigma}_i\rangle$ as the $i$th spin of the decoding/restoration process.

The most important quantity in selective freezing is the overlap of the decoded/restored bit $\mathrm{sgn}\langle\tilde{\sigma}_i\rangle$ and the original bit $\xi_i$ averaged over the output probability and the spin distribution. This is given by

$$M_{\mathrm{sf}} = \sum_\xi\prod\int dJ\prod\int d\tau P_s(\{\xi\})P_{\mathrm{out}}(\{J\},\{\tau\}|\{\xi\})\xi_i\mathrm{sgn}\langle\tilde{\sigma}_i\rangle. \qquad (7)$$

Following [3], we can prove that selective freezing cannot outperform the single-stage process if the hyperparameters can be estimated precisely. However, the purpose of selective freezing is rather to provide a relatively stable performance when the hyperparameters cannot be estimated precisely.

## 3  Modeling error-correcting codes

Let us now suppose that the output of the transmission channel consists of only the set of $p$-spin interactions $\{J_{i_1\cdots i_p}\}$. Then $h = 0$ in the Hamiltonian (5), and we set $\beta_m = 0$ for the case that all messages are equally probable. Analytical solutions are available for the infinite-range model in which the exchange interactions are present for all possible pairs of sites. Consider the noise model in which $J_{i_1\cdots i_p}$ is Gaussian with mean $p!j_0\xi_{i_1}\cdots\xi_{i_p}/N^{p-1}$ and variance $p!J^2/2N^{p-1}$. We can apply a gauge transformation $\sigma_i \to \sigma_i\xi_i$ and $J_{i_1\cdots i_p} \to J_{i_1\cdots i_p}\xi_{i_1}\cdots\xi_{i_p}$, and arrive at an equivalent $p$-spin model with a ferromagnetic bias, where

$$P(J_{i_1\cdots i_p}) = \left(\frac{N^{p-1}}{\pi J^2 p!}\right)^{1/2}\exp\left[-\frac{N^{p-1}}{J^2 p!}\left(J_{i_1\cdots i_p} - \frac{p!}{N^{p-1}}j_0\right)^2\right]. \qquad (8)$$

The infinite-range model is exactly solvable using the cavity method [8]. The method uses a self-consistency argument to consider what happens when a spin is added or removed from the system. The central quantity in this method is the

*cavity field*, which is the local field of a spin when it is added to the system, assuming that the exchange couplings act only one-way from the system to the new spin (but not from the spin back to the system). Since the exchange couplings feeding the new spin have no correlations with the system, the cavity field becomes a Gaussian variable in the limit of large valency.

The thermal average of a spin, say spin 1, is given by

$$\langle \sigma_1 \rangle = \tanh \beta h_1, \tag{9}$$

where $h_1$ is the cavity field obeying a Gaussian distribution, whose mean and variance are $p j_0 m^{p-1}$ and $p J^2 q^{p-1}/2$ respectively, where $m$ and $q$ are the magnetization and Edwards-Anderson order parameter respectively, given by

$$m \equiv \frac{1}{N} \sum_i \langle \sigma_i \rangle \quad \text{and} \quad q \equiv \frac{1}{N} \sum_i \langle \sigma_i \rangle^2. \tag{10}$$

Applying self-consistently the cavity argument to all terms in Eq. (10), we can obtain self-consistent equations for $m$ and $q$.

Now we consider selective freezing. If we introduce a freezing threshold $\theta$ so that all spins with $\langle \sigma_i \rangle^2 > \theta^2$ are frozen, then the freezing fraction $f$ is given by

$$f \equiv \frac{1}{N} \sum_i \Theta \left( \langle \sigma_i \rangle^2 - \theta^2 \right). \tag{11}$$

The thermal average of a dynamic spin in the second stage is related to the cavity fields in both stages, say, for spin 1,

$$\langle \tilde{\sigma}_1 \rangle = \tanh \beta \left\{ \tilde{h}_1 + \frac{p}{2}(p-1) J^2 r^{p-2} \chi_{tr} \tanh \beta h_1 \right\}, \tag{12}$$

where $\tilde{h}_1$ is the cavity field in the second stage, $r$ is the order parameter describing the spin correlations of the two thermodynamic stages:

$$r \equiv \frac{1}{N} \sum_i \langle \sigma_i \rangle \left\{ \langle \tilde{\sigma}_i \rangle \Theta \left[ \theta^2 - \langle \sigma_i \rangle^2 \right] + \text{sgn} \langle \sigma_i \rangle \Theta \left[ \langle \sigma_i \rangle^2 - \theta^2 \right] \right\}, \tag{13}$$

$\chi_{tr}$ is the trans-susceptibility which describes the response of a spin in the second stage to variations of the cavity field in the first stage, namely

$$\chi_{tr} \equiv \frac{1}{N} \sum_i \frac{\partial \langle \tilde{\sigma}_i \rangle}{\partial h_i}. \tag{14}$$

The cavity field $\tilde{h}_1$ is a Gaussian variable. Its mean and variance are $p j_0 \tilde{m}^{p-1}$ and $p J^2 \tilde{q}^{p-1}/2$ respectively, where $\tilde{m}$ and $\tilde{q}$ are the magnetization and Edwards-Anderson order parameter respectively, given by

$$\tilde{m} \equiv \frac{1}{N} \sum_i \left[ \Theta(\theta^2 - \langle \sigma_i \rangle^2) \langle \tilde{\sigma}_i \rangle + \Theta(\langle \sigma_i \rangle^2 - \theta^2) \text{sgn} \langle \sigma_i \rangle \right], \tag{15}$$

$$\tilde{q} \equiv \frac{1}{N} \sum_i \left[ \Theta(\theta^2 - \langle \sigma_i \rangle^2) \langle \tilde{\sigma}_i \rangle^2 + \Theta(\langle \sigma_i \rangle^2 - \theta^2) \right]. \tag{16}$$

Furthermore, the covariance between $h_1$ and $\tilde{h}_1$ is $p J^2 r^{p-1}/2$, where $r$ is given in Eq. (13). Applying self-consistently the same cavity argument to all terms in Eqs. (15), (16), (13) and (14), we arrive at the self-consistent equations for $\tilde{m}$, $\tilde{q}$, $r$ and $\chi_{tr}$. The performance of selective freezing is measured by

$$M_{\text{sf}} \equiv \frac{1}{N} \sum_i \left[ \Theta(\theta^2 - \langle \sigma_i \rangle^2) \text{sgn} \langle \tilde{\sigma}_i \rangle + \Theta(\langle \sigma_i \rangle^2 - \theta^2) \text{sgn} \langle \sigma_i \rangle \right]. \tag{17}$$

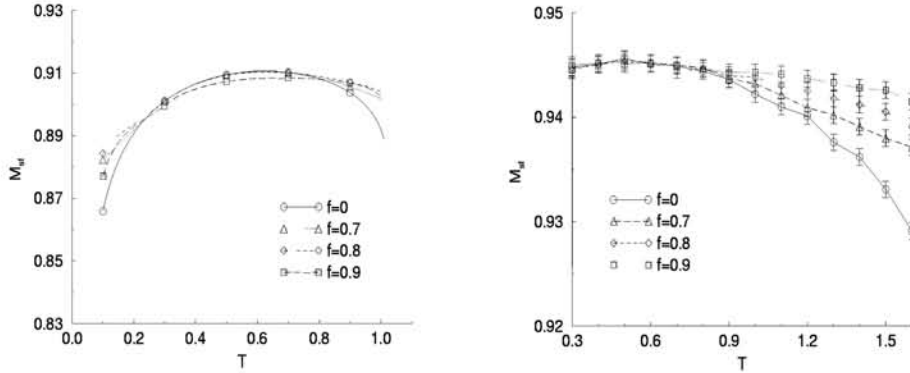

Figure 1: The overlap $M_{\mathrm{sf}}$ as a function of the decoding temperature $T$ for various given values of freezing fraction $f$. In this and the following figure, $f = 0$ corresponds to one-stage decoding/restoration. (a) Theoretical results for $p = 3$, $j_0 = 0.8$ and $J = 1$; (b) results of Monte Carlo simulations for $p = 2$ and $j_0 = J = 1$.

In the example in Fig. 1(a), the overlap of the single-stage dynamics reaches its maximum at the Nishimori point $T_N = J^2/2j_0$ as expected. We observe that the tolerance against variations in $T$ is enhanced by selective freezing both above and below the optimal temperature (see especially $f = 0.8$). This shows that the region of advantage for selective freezing is even broader than that discussed in [7], where improvement is only observed above the optimal temperature.

The advantages of selective freezing are confirmed by Monte Carlo simulations shown in Fig. 1(b). For one-stage dynamics, the overlap is maximum at the Nishimori point ($T_N = 0.5$) as expected. However, it deterriorates rather rapidly when the decoding temperature increases. In contrast, selective freezing maintains a more steady performance, especially when $f = 0.9$.

## 4   Modeling image restoration

In conventional image restoration problems, a given degraded image consists of the set of pixels $\{\tau_i\}$, but not the set of exchange interactions $\{J_{i_1, \ldots, i_p}\}$. In this case, $\beta = 0$ in the Hamiltonian (5). The pixels $\tau_i$ are the degraded versions of the source pixels $\xi_i$, corrupted by noise which, for convenience, is assumed to be Gaussian with mean $a\xi_i$ and variance $\tau^2$. In turn, the source pixels satisfy the prior distribution in Eq. (1) for smooth images.

Analysis of the mean-field model with extensive valency shows that selective freezing performs as well as one-stage dynamics, but cannot outperform it. Nevertheless, selective freezing provides a rather stable performance when the hyperparameters cannot be estimated precisely. Hence we model a situation common in modern communication channels carrying multimedia traffic, which are often bursty in nature. Since burstiness results in intermittent interferences, we consider a distribution of the degraded pixels with two Gaussian components, each with its own characteristics,

$$P(\tau_i|\xi_i) = f_1 \frac{\exp\left[-\frac{1}{2\tau_1^2}(\tau_i - a_1\xi_i)^2\right]}{\sqrt{2\pi\tau_1^2}} + (1 - f_1) \frac{\exp\left[-\frac{1}{2\tau_2^2}(\tau_i - a_2\xi_i)^2\right]}{\sqrt{2\pi\tau_2^2}}. \quad (18)$$

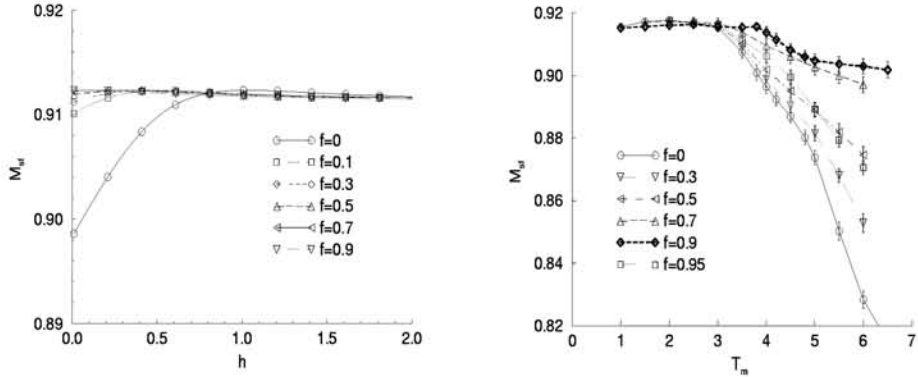

Figure 2: (a) The performance of selective freezing with 2 components of Gaussian noise at $\beta_s = 1.05$, $f_1 = 4f_2 = 0.8$, $a_1 = 5a_2 = 1$ and $\tau_1 = \tau_2 = 0.3$, The restoration agent operates at the optimal ratio $\beta_m/h$ which assumes a single noise component with the overall mean 0.84 and variance 0.4024. (b) Results of Monte Carlo simulations for the overlaps of selective freezing compared with that of the one-stage dynamics for two-dimensional images generated at the source prior temperature $T_s = 2.15$.

Suppose the restoration agent operates at the optimal ratio of $\beta_m/h$ which assumes a single noise component. Then there will be a degradation of the quality of the restored images. In the example in Fig. 2(a), the reduction of the overlap $M_{sf}$ for selective freezing is much more modest than the one-stage process $(f = 0)$. Other cases of interest, in which the restoration agent operates on other imprecise estimations, are discussed in [7]. All confirm the robustness of selective freezing.

It is interesting to study the more realistic case of two-dimensional images, since we have so far presented analytical results for the mean-field model only. As confirmed by the results for Monte carlo simulations in Fig. 2(b), the overlaps of selective freezing are much more steadier than that of the one-stage dynamics when the decoding temperature changes. This steadiness is most remarkable for a freezing fraction of $f = 0.9$.

## 5   Discussions

We have introduced a multistage technique for error-correcting codes and image restoration, in which the information extracted from the former stage can be used selectively to improve the performance of the latter one. While the overlap $M_{sf}$ of the selective freezing is bounded by the optimal performance of the one-stage dynamics derived in [3], it has the advantage of being tolerant to uncertainties in hyperparameter estimation. This is confirmed by both analytical and simulational results for mean-field and finite-dimensional models. Improvement is observed both above and below the optimal decoding temperature, superseding the observations in [7]. As an example, we have illustrated its advantage of robustness when the noise distribution is composed of more than one Gaussian components, such as in the case of modern communication channels supporting multimedia applications.

Selective freezing can be generalized to more than two stages, in which spins that remain relatively stable in one stage are progressively frozen in the following one.

It is expected that the performance can be even more robust.

On the other hand, we have a remark about the basic assumption of the cavity method, namely that the addition or removal of a spin causes a small change in the system describable by a perturbative approach. In fact, adding or removing a spin may cause the thermal averages of other spins to change from below to above the thresholds $\pm\theta$ (or vice versa). This change, though often small, induces a non-negligible change of the thermal averages from fractional values to the frozen values of $\pm 1$ (or vice versa) in the second stage. The perturbative analysis of these changes is only approximate. The situation is reminiscent of similar instabilities in other disordered systems such as the perceptron, and are equivalent to Almeida-Thouless instabilities in the replica method [9]. A full treatment of the problem would require the introduction of a rough energy landscape [9], or the replica symmetry breaking ansatz in the replica method [8]. Nevertheless, previous experiences on disordered systems showed that the corrections made by a more complete treatment may not be too large in the ordered phase. For example, simulational results in Figs. 1(b) are close to the corresponding analytical results in [7].

In practical implementations of error-correcting codes, algorithms based on belief-propagation methods are often employed [10]. It has recently been shown that such decoded messages converge to the solutions of the TAP equations in the corresponding thermodynamic system [11]. Again, the performance of these algorithms are sensitive to the estimation of hyperparameters. We propose that the selective freezing procedure has the potential to make these algorithms more robust.

### Acknowledgments

This work was partially supported by the Research Grant Council of Hong Kong (HKUST6157/99P).

# References

[1] R. J. McEliece, *The Theory of Information and Coding*, Encyclopedia of Mathematics and its Applications (Addison-Wesley, Reading, MA 1977).

[2] S. Geman and D. Geman, IEEE Trans. PAMI **6**, 721 (1984).

[3] H. Nishimori and K. Y. M. Wong, Phys. Rev. E **60**, 132 (1999).

[4] D. J. C. Mackay, Neural Computation **4**, 415 (1992).

[5] J. M. Pryce and A. D. Bruce, J. Phys. A **28**, 511 (1995).

[6] K. Y. M. Wong, Europhys. Lett. **36**, 631 (1996).

[7] K. Y. M. Wong and H. Nishimori, submitted to Phys. Rev. E (2000).

[8] M. Mézard, G. Parisi, and V.A. Virasoro, *Spin Glass Theory and Beyond* (World Scientific, Singapore 1987).

[9] K. Y. M. Wong, Advances in Neural Information Processing Systems **9**, 302 (1997).

[10] B. J. Frey, *Graphical Models for Machine Learning and Digital Communication* (MIT Press, 1998).

[11] Y. Kabashima and D. Saad, Europhys. Lett. **44**, 668 (1998).
